# Coupling Nonparametric Mixtures via Latent Dirichlet Processes

**Dahua Lin**
MIT CSAIL
dhlin@mit.edu

**John Fisher**
MIT CSAIL
fisher@csail.mit.edu

## Abstract

Mixture distributions are often used to model complex data. In this paper, we develop a new method that jointly estimates mixture models over multiple data sets by exploiting the statistical dependencies between them. Specifically, we introduce a set of latent Dirichlet processes as sources of component models (atoms), and for each data set, we construct a nonparametric mixture model by combining sub-sampled versions of the latent DPs. Each mixture model may acquire atoms from different latent DPs, while each atom may be shared by multiple mixtures. This multi-to-multi association distinguishes the proposed method from previous ones that require the model structure to be a tree or a chain, allowing more flexible designs. We also derive a sampling algorithm that jointly infers the model parameters and present experiments on both document analysis and image modeling.

## 1 Introduction

Mixture distributions have been widely used for statistical modeling of complex data. Classical formulations specify the number of components a priori, leading to difficulties in situations where the number is either unknown or hard to estimate in advance. Bayesian nonparametric models, notably those based on Dirichlet processes (DPs) [14, 16], have emerged as an important method to address this issue. The basic idea of DP mixture models is to use a sample of a DP, which is itself a distribution over a countably infinite set, as the prior for component parameters.

One significant assumption underlying a DP mixture model is that observations are *infinitely exchangeable*. This assumption does not hold in the cases with multiple groups of data, where samples in different groups are generally not exchangeable. Among various approaches to this issue, hierarchical Dirichlet processes (HDPs) [20], which organize DPs into a tree with parents acting as the base measure for children, is one of the most popular. HDPs have been extended in a variety of ways. Kim and Smyth [9] incorporated group-specific random perturbations, allowing component parameters to vary across different groups. Ren *et al.* [17] proposed dynamic HDPs, which combine the DP at a previous time step with a new one at the current time step.

Other methods have also been developed. MacEachern [13] proposed a DDP model that allows parameters to vary following a stochastic process. Griffin and Steel [6] proposed the order-based DDP, where atoms can be weighted differently via the permutation of the Beta variables for stick-breaking. Chung and Dunson [3] carried this approach further, using local predictors to select subsets of atoms. Recently, the connections between Poisson, Gamma, and Dirichlet processes have been exploited. Rao and Teh [15] proposed the spatially normalized Gamma process, where a set of dependent DPs can be derived by normalizing restricted projections of an auxiliary Gamma process over overlapping sub-regions. Lin et al [12] proposed a new construction of dependent DPs, which supports dynamic evolution of a DP through operations on the underlying Poisson processes.

Our primary goal here is to describe multiple groups of data through coupled mixture models. Sharing statistical properties across different groups allows for more reliable model estimation, especially

when the observed samples in each group are limited or noisy. From a probabilistic standpoint, this framework can be obtained by devising a joint stochastic process that generates DPs with mutual dependency. Particularly, it is desirable to have a design that satisfies three properties: (1) Sharing of mixture components (atoms) between groups. (2) The marginal distribution of atoms for each group remains a DP. (3) Flexible configuration of inter-group dependencies. For example, the prior weight of a common atom can vary across groups.

Achieving these goals simultaneously is nontrivial. Whereas several existing constructions [3, 6, 12, 15] meet the first two properties, they impose restrictions on the model structure (*e.g.* the groups need to be arranged into a tree or a chain). We present a new framework to address this issue. Specifically, we express mixture models for each group as a stochastic combination over a set of *latent DPs*. The multi-to-multi association between data groups and latent DPs provides much greater flexibility to model configurations, as opposed to prior work (we provide a detailed comparison in section 3.2). We also derive an MCMC sampling method to infer model parameters from grouped observations.

## 2  Background

We provide a review of Dirichlet processes in order to lay the theoretical foundations of the method described herein. We also discuss the related construction of dependent DPs proposed by [12], which exploits the connection between Poisson and Dirichlet processes to support various operations.

A Dirichlet process, denoted by $\mathrm{DP}(\alpha B)$, is a distribution over probability measures, which is characterized by a *concentration parameter* $\alpha$ and a *base measure* $B$ over an underlying space $\Omega$. Each sample path $D \sim \mathrm{DP}(\alpha B)$ is itself a distribution over $\Omega$. Sethuraman [18] showed that $D$ is almost surely discrete (with countably infinite support), and can be expressed as

$$D = \sum_{k=1}^{\infty} \pi_k \delta_{\phi_k}, \quad \text{with } \pi_k = v_k \prod_{l=1}^{k-1}(1 - v_l), \ v_k \sim \mathrm{Beta}(1, \alpha). \tag{1}$$

This is known as the *stick breaking representation* of a DP. This discrete nature makes a DP particularly suited to serve as a prior for component parameters in mixture models.

Generally, in a DP mixture model, each data sample $x_i$ is considered to be generated from a component model with parameter $\theta_i$, denoted by $\mathcal{G}(\theta_i)$. The component parameters are samples from $D$, which is itself a realization of a DP. The formulation is given below

$$D \sim \mathrm{DP}(\alpha B), \quad \theta_i \sim D, \ x_i \sim \mathcal{G}(\theta_i), \ \text{for } i = 1, \dots, n. \tag{2}$$

As $D$ is an infinite series, it is infeasible to instantiate $D$. As such, the *Chinese restaurant process*, given by Eq. 3, is often used to directly sample the component parameters, with $D$ integrated out.

$$p(\theta_i | \boldsymbol{\theta}_{/i}) = \sum_{k=1}^{K_{/i}} \frac{m_{/i}(k)}{\alpha + (n-1)} \delta_{\phi_k} + \frac{\alpha}{\alpha + (n-1)} B. \tag{3}$$

Here, $\boldsymbol{\theta}_{/i}$ denotes all component parameters except $\theta_i$, $K_{/i}$ denotes the number of distinct atoms among them, and $m_{/i}(k)$ denotes the number of occurrences of the atom $\phi_k$. When $x_i$ is given, the likelihood to generate $x_i$ conditioned on $\theta_i$ can be incorporated, resulting in an modulated sampling scheme described below. Let $f(x_i; \phi)$ denote the likelihood to generate $x_i$ *w.r.t.* $\mathcal{G}(\phi)$, and $f(x_i; B)$ denote the marginal likelihood *w.r.t.* the parameter prior $B$. Then, with a probability proportional to $m_{/i}(k)f(x_i; \phi_k)$, we set $\theta_i = \phi_k$, and with a probability proportional to $\alpha f(x_i; B)$, we draw an new atom from $B(\cdot | x_i)$, which is the posterior parameter distribution given $x_i$.

Recently, Lin *et al.* [12] proposed a new construction of DPs based on the connections between Poisson, Gamma, and Dirichlet processes. The construction provides three operations to derive new DPs depending on existing ones, which we will use to develop the coupled DP model. Here, we provide a brief review of these operations.

**(1) Superposition.** Let $D_k \sim \mathrm{DP}(\alpha_k B_k)$ for $k = 1, \dots, K$ be independent DPs and $(c_1, \dots, c_K) \sim \mathrm{Dir}(\alpha_1, \dots, \alpha_K)$. Then the stochastic convex combination of these DPs as below remains a DP:

$$c_1 D_1 + \cdots + c_K D_K \ \sim \ \mathrm{DP}(\alpha_1 B_1 + \cdots + \alpha_K B_K). \tag{4}$$

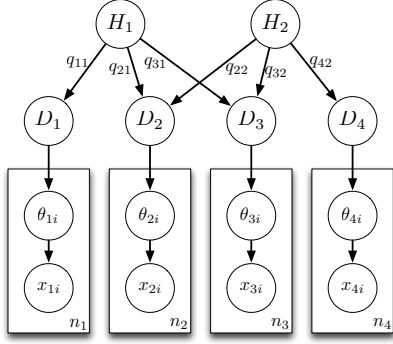

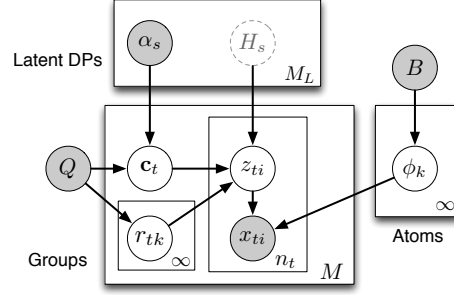

Figure 1: This shows the graphical model of the coupled DP formulation on a case with four groups and two latent DPs. Each mixture model $D_t$ inherits atoms from $H_s$ with a probability $q_{ts}$, resulting in Eq.(7).

Figure 2: The reformulated model for Gibbs sampling contains latent DPs, groups of data, and atoms. Each sample $x_{ti}$ is attached a label $z_{ti}$ that assigns it an atom $\phi_{z_{ti}}$. To generate $z_{ti}$, we draw a latent DP (from $\mathrm{Mult}(\mathbf{c}_t)$) and choose a label therefrom. In sampling, $H_s$ is integrated out, resulting in mutual dependency between $z_{ti}$, as in the Chinese restaurant process.

**(2) Sub-sampling.** Let $D = \sum_{k=1}^{\infty} \pi_k \delta_{\phi_k} \sim DP(\alpha B)$. One obtains a new DP by sub-sampling $D$ via independent Bernoulli trials. Given a *sub-sampling probability* $q$, one draws a binary value $r_k$ with $\Pr(r_k = 1) = q$ for each atom $\phi_k$ to decide whether to retain it, resulting in a DP as

$$S_q(D) \triangleq \sum_{k:r_k=1} \pi'_k \delta_{\phi_k} \sim DP(\alpha q B). \tag{5}$$

Here, $S_q$ denotes the sub-sampling operation (with probability $q$), and $\pi'_k$ is the re-normalized coefficient for $\phi_k$, which is given by $\pi'_k = \pi_k / \sum_k r_k \pi_k$.

**(3) Transition.** Given $D = \sum_{k=1}^{\infty} \pi_k \delta_{\phi_k} \sim DP(\alpha B)$, perturbing the locations of each atom following a probabilistic transition kernel $T$ also yields a new DP, given by $T(D) \triangleq \sum_{k=1}^{\infty} \pi_k \delta_{T(\phi_k)}$.

While these operations were originally developed to evolve a DP along a Markov chain, we show in the next section that they can also be utilized to construct models with different structures.

## 3 Coupled Nonparametric Mixture Models

Our primary goal is to develop a joint formulation over group-wise DP mixture models where components are shared across different groups and the weights and parameters of shared components vary across groups. We propose a new construction illustrated in Figure 1. Suppose there are $M$ groups of data, each with a mixture model. They are coupled by $M_L$ *latent DPs*. The generative formulation is then described as follows: First, generate $M_L$ latent DPs independently, as

$$H_s \sim DP(\alpha_s B), \quad \text{for } s = 1, \ldots, M_L. \tag{6}$$

Second, generate $M$ dependent DPs, each for a group of data, by combining the sub-sampled versions of the latent DPs through stochastic convex combination. For each $t = 1, \ldots, M$,

$$D_t = \sum_{s=1}^{M_L} c_{ts} S_{q_{ts}}(H_s), \quad \text{with } (c_{t1}, \ldots, c_{tM_L}) \sim \mathrm{Dir}(\alpha_1 q_{t1}, \ldots, \alpha_{M_L} q_{tM_L}). \tag{7}$$

Intuitively, for each group of data (say the $t$-th), we choose a subset of atoms from each latent source and bring them together to generate $D_t$. Here, $q_{ts}$ is the prior probability that an atom in $H_s$ will be inherited by $D_t$. Note that this formulation can be further extended into $D_t = \sum_s c_{ts} T_t(S_{q_{ts}}(H_s))$. Here, $T_t$ is a probabilistic transition kernel. Using the transition operation, this extension allows parameters to vary across different groups. Particularly, the atom parameter would be an adapted version from $T_t(\phi_k, \cdot)$ instead of $\phi_k$ itself, when the atom $\phi_k$ is inherited by $D_t$.

Third, generate the component parameters and data samples in the standard way, as

$$\theta_{t,i}|D_t \sim D_t, \text{ and } x_{t,i}|\theta_{t,i} \sim \mathcal{G}(\theta_{t,i}), \quad \text{for } i = 1, \ldots, n_t, \ t = 1, \ldots, M. \tag{8}$$

Here, $x_{t,i}$ is the $i$-th data sample in the $t$-th group, and $\theta_{t,i}$ is the associated atom parameter.

### 3.1 Theoretical Analysis

The following theorems (proofs provided in supplementary material) demonstrate that, as a result of the construction above, the marginal distribution of $D_t$ is a DP:

**Theorem 1.** *The stochastic process $D_t$ given by Eq.(7) has $D_t \sim \mathrm{DP}(\beta_t B)$, with $\beta_t = \sum_{s=1}^{M_L} \alpha_s q_{ts}$.*

We also show that they are dependent, with the covariance given by the theorem below.

**Theorem 2.** *Let $t_1 \neq t_2$ and $U$ be a measurable subset of $\Omega$, then*

$$\mathrm{Cov}(D_{t_1}(U), D_{t_2}(U)) = \frac{1}{\beta_{t_1}\beta_{t_2}} \sum_{s=1}^{M_L} \frac{(\alpha_s q_{t_1 s} q_{t_2 s})^2}{\alpha_s q_{t_1 s} q_{t_2 s} + 1} B(U)(1 - B(U)). \tag{9}$$

It can be seen that the hyper-parameters influence the model characteristics in different ways. The inheritance probabilities (*i.e.* the $q$-values) control how closely the models are coupled. Two models are strongly coupled, if there exists a subset of latent DPs, from which both inherit atoms with high probabilities, while their coupling is much weaker if the associated $q$-values are set differently. The latent concentration parameters (*i.e.* the values of $\alpha_s$) control how frequently new atoms are created. Generally, higher values of $\alpha_s$ lead to more atoms being associated with the data, resulting in finer clusters. Another important factor is $M_L$, the number of latent DPs. A large number of latent DPs provides fine-grained control of the model configuration at the cost of increased complexity.

### 3.2 Comparison with Other Models

We review related approaches and discuss their differences with the one proposed here. Similar to this work, HDPs [20] model grouped data. Such models must be arranged into a tree, *i.e.* each child can only have one parent. Our model allows the mixture model for each group to inherit from multiple sources, making it applicable to more general contexts.

It is worth emphasizing that enabling inheritance from multiple parents is not just a straightforward extension, as it entails both theoretical and practical challenges: First, to combine atoms from multiple DPs while guaranteeing that the resultant process remains a DP requires careful design of the formulation (*e.g.* the combination coefficients should be from a Dirichlet distribution, and each parent DP should be properly sub-sampled). Second, the sampling procedure has to determine the source of each atom, which, again, is nontrivial and needs special algorithmic design (see section 4) to maintain the detailed balance.

SNΓP [15] defines a gamma process $G$ over an extended space. For each group $t$, a DP $D_t$ is derived through normalized restriction of $G$ into a measurable subset. The DPs derived on overlapped subsets are dependent. Though motivated differently, this construction can be reduced to a formulation in the form $D_t = \sum_{j \in R_t} c_{tj} H_j$, where $R_t$ is the subset of latent DPs used for $D_t$. Compared to Eq.(7), we can see that it is essentially a special case of the present construction without sub-sampling (*i.e.* all $q$-values equal 1). Consequently, the combination coefficients have to satisfy $(c_{tj})_{j \in R_t} \sim \mathrm{Dir}((\alpha_j)_{j \in R_t})$, implying that the relative weights of two latent sources are restricted to be the same in all groups that inherit from both. In contrast, the approach here allows the weights of latent DPs to vary across groups. Also, SNΓP doesn't allow atom parameters to vary across groups.

## 4 Sampling Algorithm

This section introduces a Gibbs sampling algorithm to jointly estimate the mixture models of multiple groups. Overall, this algorithm is an extension to the Chinese restaurant process, with several new aspects: (1) The conditional probability of labels depend on the total number of samples associated with it over the entire corpus (instead of that within a specific group). Note that it also differs

from HDP, where such probabilities depend on the number of associated *tables*. (2) Each group maintains a distribution over the latent DPs to choose from, which reflects the different contributions of these sources. (3) It leverages the sub-sampling operation to explicitly control the model complexity. In particular, each group maintains indicators of whether particular atoms are inherited, and as a consequence, the ones that are deemed irrelevant are put out of scope. (4) As there are multiple latent DPs, for each atom, there is uncertainty about where it comes from. We have a specific step that takes this into account, which allows reassigning an atom to different sources.

We first set up the notations. Recall that there are $M$ groups of data, and $M_L$ latent DPs to link between them. The observations in the $t$-th group are $x_{t1}, \ldots, x_{tn_t}$. We use $\phi_k$ to denote an atom. Note here that the index $k$ is a globally unique identifier of the atom, which would not be changed during atom relocation. Since an atom may correspond to multiple data samples. Instead of instantiating the parameter $\theta_{ti}$ for each data sample $x_{ti}$, we attach to $x_{ti}$ an indicator $z_{ti}$ that associates the sample to a particular atom. This is equivalent to setting $\theta_{ti} = \phi_{z_{ti}}$. To facilitate the sampling process, for each atom $\phi_k$, we maintain an indicator $s_k$ specifying the latent DP that contains it, and a set of counters $\{m_{tk}\}$, where $m_{tk}$ equals the number of associated data samples in $t$-th group. We also maintain a set $I_s$ for $H_s$ (the $s$-th latent DP), which contains the indices of all atoms therein.

The model in Eq.(7) and (8) can then be reformulated, as shown in Fig 2. It consists of four steps: **(1) Generate latent DPs:** for each $s = 1, \ldots, M_L$, we draw $H_s \sim \mathrm{DP}(\alpha_s B)$. **(2) Generate the combination coefficients:** for each group $t$, we draw $(c_{t1}, \ldots, c_{tM_L}) \sim \mathrm{Dir}(\alpha_1 q_{t1}, \ldots, \alpha_{M_L} q_{tM_L})$, which gives the group-specific prior over the sources for the $t$-th group. **(3) Decide inheritance:** for each atom $\phi_k$, we draw a binary variable $r_{tk}$ with $\Pr(r_{tk} = 1) = q_{ts_k}$ to indicate whether $\phi_k$ is inherited by the $t$-th group. Here $s_k$ is the index of the latent DP which $\phi_k$ is from. **(4) Generate data:** to generate $x_{ti}$, we first choose a latent DP by drawing $u \sim \mathrm{Mult}(c_{t1}, \ldots, c_{tM_L})$, then draw an atom from $H_u$, using it to produce $x_{ti}$. Based on this formulation, we derive the following Gibbs sampling steps to update the atom parameters and other hidden variables.

**(1) Update labels.** Recall that each data sample $x_{ti}$ is associated with a label variable $z_{ti}$ that indicates the atom accounting for $x_{ti}$. To draw $z_{ti}$, we first have to choose a particular latent DP as the source (we denote the index of this DP by $u_{ti}$). Let $\mathbf{z}_{/ti}$ denote all labels except $z_{ti}$, and $\mathbf{r}_t$ denote the inheritance indicators. Then, we get the likelihood of $x_{ti}$ (with $H_s$ integrated out) as

$$p(x_{ti}|u_{ti} = s, \mathbf{r}_t, \mathbf{z}_{/ti}) = \frac{1}{w_{st/i} + q_{ts}\alpha_s} \left( \sum_{k \in I_s : r_{tk} = 1} m_{*k/ti} f(x_{ti}; \phi_k) + q_{ts}\alpha_s f(x_{ti}; B) \right). \quad (10)$$

Here, $m_{*k/ti}$ is the total number of samples associated with $\phi_k$ in all groups (except for $x_{ti}$), $w_{st/i} = \sum_{k \in I_s : r_{tk} = 1} m_{*k/ti}$, $f(x_{ti}; \phi_k)$ is the pdf at $x_{ti}$ w.r.t. $\phi_k$, and $f(x_{ti}; B) = \int_\theta f(x_{ti}; \theta) B(\theta) d\theta$. Derivations of this and other formulas for sampling are in the supplemental document. Hence,

$$p(u_{ti} = s|\text{others}) \propto p(u_{ti} = s|\mathbf{c}_t) p(x_{ti}|u_{ti} = s, \mathbf{z}_{/ti}) = c_{ts} p(x_{ti}|u_{ti} = s, \mathbf{z}_{/ti}). \quad (11)$$

Here, $\mathbf{c}_t = (c_{t1}, \ldots, c_{tM_L})$ are the group-specific prior over latent sources. Once a latent DP is chosen (using the formula above), we can then draw a particular atom. This is similar to the Chinese restaurant process: with a probability proportional to $m_{*k/ti} f(x_{ti}; \phi_k)$, we set $z_{ti} = k$, and with a probability proportional to $q_{ts}\alpha_s f(x_{ti}; B)$, we draw a new atom from $B(\cdot|x_i)$. Only the atoms that is contained in $H_s$ and has $r_{tk} = 1$ (inherited by $D_t$) can be drawn at this step.

We have to modify relevant quantities accordingly, such as $m_{tk}$, $w_s$, and $I_s$, when a label $z_{ti}$ is changed. Moreover, when a new atom $\phi_k$ is created, it will be initially assigned to the latent DP that generates it (*i.e.* setting $s_k = u_{ti}$).

**(2) Update inheritance indicators.** If an atom $\phi_k$ is associated with some data in the $t$-th group, then we know for sure that it is inherited by $D_t$, and thus we can set $r_{tk} = 1$. However, if $\phi_k$ is not observed, it doesn't imply $r_{tk} = 0$. For such an atom (suppose it is from $H_s$), we have

$$\frac{\Pr(r_{tk} = 1|\text{others})}{\Pr(r_{tk} = 0|\text{others})} = \frac{q_{ts} \cdot p(\mathbf{z}_t|r_{tk} = 1, \text{others})}{(1 - q_{ts}) \cdot p(\mathbf{z}_t|r_{tk} = 0, \text{others})} = \frac{q_{ts}}{1 - q_{ts}} \frac{\gamma(\tau_{s/t}, n_t)}{\gamma(\tau_{s/t} + m_{*k/t}, n_t)}. \quad (12)$$

Here, $\tau_{s/t} = q_{ts}\alpha_s + \sum_{k' \in I_s - \{k\}} m_{*k'/t}$ and $m_{*k'/t}$ is the number of samples associated with $k'$ in all other groups (excluding the ones in the $t$-th group). $\gamma$ is a function defined by $\gamma(\tau, n) = \prod_{i=0}^{n-1}(\tau + i) = \Gamma(\tau + n)/\Gamma(\tau)$. Intuitively, when $m_{*k}$ is large (indicating that $\phi_k$ appears frequently

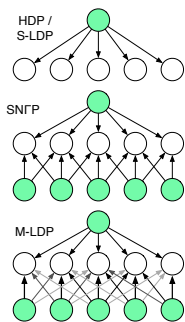

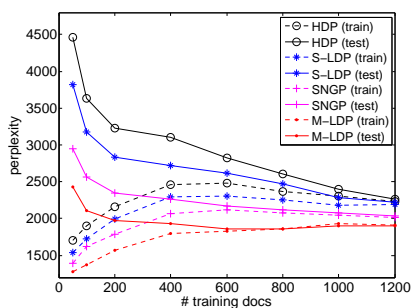

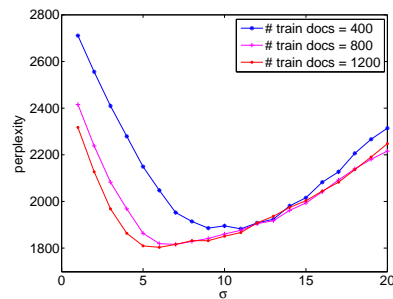

Figure 3: model structures.

Figure 4: The results on NIPS data obtained with training sets of different sizes.

Figure 5: The results on NIPS data using M-LDP, with different $\sigma$ values.

in other groups) or $n_t$ is large, $\phi_k$ is likely to appear in the $t$-th group if it is inherited. Under such circumstances, if $\phi_k$ not seen, then it is probably not inherited.

**(3) Update combination coefficients.** The coefficients $\mathbf{c}_t = (c_{t1}, \ldots, c_{tM_L})$ reflect the relative contribution of each latent DP to the $t$-th group. $\mathbf{c}_t$ follows a Dirichlet distribution a priori (see Eq.(7)). Given $\mathbf{z}_t$, the labels of all samples in the $t$-th group, we have

$$\mathbf{c}_t | \mathbf{z}_t \sim \mathrm{Dir}\left(\alpha_1 q_{t1} + \sum_{k \in I_1} m_{tk}, \ldots, \alpha_{M_L} q_{tM_L} + \sum_{k \in I_{M_L}} m_{tk}\right). \tag{13}$$

Here, $\sum_{k \in I_s} m_{tk}$ is the total number of samples in the $t$-th group that are associated with $H_s$.

**(4) Update atom parameters.** Given all the labels, we can update the atoms, by re-drawing their parameters from the posterior distributions. Let $X_k$ denote the set of all data samples associated with the $k$-th atom, then we can draw $\phi_k \sim B(\cdot|X_k)$, where $B(\cdot|X_k)$ denotes the posterior distribution conditioned on $X_k$, with the pdf given by $B(\phi|X_k) \propto B(\phi) \prod_{x \in X_k} f(x_k; \phi)$.

**(5) Reassign atoms.** In this model, each atom is almost surely from a unique latent DP (*i.e.* it never comes from two distinct sources). This leads to an important question: *How to we assign atoms to latent DPs?* Initially, an atom is assigned to the latent DP from which it is generated. This is not necessarily optimal. Here, we treat the assignment of each atom as a variable. Consider an atom $\phi_k$, with $s_k$ indicating its corresponding source DP. Then, we have

$$p(s_k = j|\text{others}) = \prod_{t:r_{tk}=1} q_{ts} \prod_{t:r_{tk}=0} (1 - q_{ts}). \tag{14}$$

When an atom $\phi_k$ that was in $H_s$ is reassigned to $H_{s'}$, we have to move the index $k$ from $I_s$ to $I_{s'}$.

## 5 Experiments

The framework developed in this paper provides a generic tool to model grouped data. In this section, we present experiments on two applications: *document analysis* and *scene modeling*. The primary goal is to demonstrate the key distinctions between the proposed approach and other non-parametric methods, and study how the new design influences empirical performance.

### 5.1 Document Analysis

Topic models [1, 2, 7, 20] have been widely used for statistical analysis of documents. In general, a topic model comprises a set of *topics*, each associated with a multinomial distribution, from which words can be independently generated. Here, we formulate a *Coupled Topic Model* by extending LDA [2] to model multiple groups of documents. Specifically, it associates the $t$-th group with a mixture of topics, characterized by a DP sample $D_t$. With this given, the words in a document are generated independently, each from a topic drawn from $D_t$. To exploit the statistical dependency between groups, we further introduce a set of latent DPs to link between these mixtures, as described

above. The NIPS (1-17) database [5], which contains $2484$ papers published from $1987$ to $2003$, is used in our experiments. We clean the data by removing the words that occur fewer than $10$ times over the corpus and those that appear in more than $2000$ papers, resulting in a reduced vocabulary comprised of $11729$ words. The data are divided into $17$ groups, one for each year.

We perform experiments on several configurations, with different ways to connect between latent sources and data groups, as illustrated in Figure 3. (1) *Single Latent DP (S-LDP)*: there is only one latent DP connecting to all groups, with $q$-values set to $0.5$. Though with a structure similar to HDP, the formulation is actually different: HDP generates group-specific mixtures by using the latent DP as the base measure, while our model involves explicit sub-sampling. (2) *Multi Latent DP (M-LDP)*: there are two types of latent DPs – local and global ones. The local latent DPs are introduced to help sharing statistical strength among the groups close to each other, so as to capture the intuition that papers published in consecutive years are more likely to share topics than those published in distant years. The inheritance probability from a local latent DP $H_s$ to $D_t$ is set as $q_{ts} = \exp(-|t-s|/\sigma)$. Also, recognizing that some topics may be shared across the entire corpus, we also introduce a global latent DP, from which every group inherit atoms with the same probability, which allows distant groups to be connected. This design illustrates the flexibility of the proposed framework and how one can leverage this flexibility to address practical needs.

For comparison, we also consider another setting of $q$-values under the *M-LDP* structure: to set $q_{ts} = \mathbb{I}(|t-s| \le \sigma)$, that is to connect $D_t$ and $H_s$ only when $|t-s| \le \sigma$, with $q_{ts} = 1$. Under this special setting, the formulation reduces to SN$\Gamma$P [15]. We also test HDP following exactly the settings given in [20]: $\alpha_0 \sim \mathrm{Gamma}(0.1, 0.1)$ and $\gamma \sim \mathrm{Gamma}(5, 0.1)$. Other design parameters are set as below. We place a weak prior over $\alpha_s$ for each latent DP, as $\alpha_s \sim \mathrm{Gamma}(0.1, 0.1)$, and periodically update its value. The base distribution $B$ is assumed to be $\mathrm{Dir}(1)$, which is actually a uniform distribution over the probability simplex.

The first experiment is to compare different methods on training sets of various sizes. We divide all papers into two disjoint halves, respectively for training and testing. In each test, models are estimated upon a subset of specific size randomly chosen from the training corpus. The learned models are then respectively tested on the training subset and the held-out testing set, so as to study the gap between empirical and generalized performance, which is measured in terms of *perplexity*.

From Figure 4, we observe: (1) In general, as the training set size increases, the perplexity evaluated on the training set increases and that on the testing set decreases. However, such convergence is faster when local coupling is used (*e.g.* in SN$\Gamma$P and M-LDP). This suggests that the *sharing of statistical strength* through local latent DPs improves the reliability of the estimation, especially when the training data are limited. (2) Even when the training set size is increased to $1200$, the methods using local coupling still yield lower perplexity than others. This is partly ascribed to the model structure. For example, the papers published in consecutive years tend to share lots of topics, however, the topics may not be as similar when you compare papers published recently to those a decade ago. A set of local latent DPs may capture such relations more effectively than a single global one. (3) The proposed method under M-LDP setting outperforms other methods, including SN$\Gamma$P. In M-LDP, the contribution of $H_s$ to $D_t$ decreases gracefully as $|t-s|$ increases. This way encourages each latent DP to be locally focused, while allowing the atoms therein to be shared across the entire corpus. This is enabled through the use of explicit sub-sampling. The SN$\Gamma$P, instead, provides no mechanism to vary the contributions of the latent DPs, and has to make a hard limit of their spans to achieve locality. Whereas this issue could be addressed through multiple level of latent nodes with different spans, it will increase the complexity, and thus the risk of overfitting.

For M-LDP, recall that we set $q_{ts} = \exp(-|t-s|/\sigma)$. Here, $\sigma$ is an important design parameter that controls the range of local coupling. The results acquired with different $\sigma$ values are shown in Figure 5. Optimal performance is attained when the choice of $\sigma$ balances the need to share atoms and the desire to keep the latent DPs locally focused. Generally, the optimum of $\sigma$ depends on data. When the training set is limited, one may increase its value to enlarge the coupling range.

## 5.2 Scene Modeling

Scene modeling is an important task in computer vision. Among various approaches, topic models that build upon *bag-of-features* image representation [4, 11, 21] have become increasingly popular

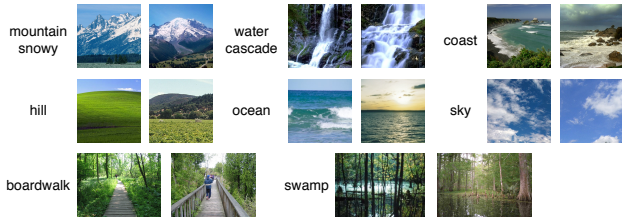

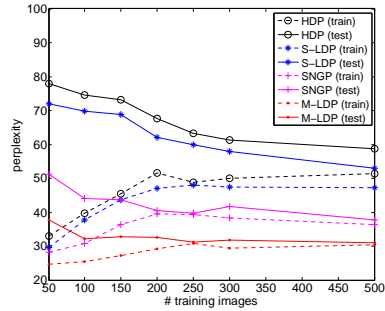

Figure 6: This figure shows example images in all eight categories selected for the experiment.

Figure 7: The results on SUN data, with training sets of different sizes.

and are widely used for statistical modeling of visual scenes. Along this trend, Dirichlet processes have also been employed to discover visual topics from observed scenes [10, 19].

We apply the proposed method to jointly model the topics in multiple scene categories. Rather than pursuing the optimal scene model, here we primarily aimed at comparing different nonparametric methods in mixture model estimation, under a reasonable setting. We choose a subset from the SUN database [22]. The selected set comprises eight outdoor categories: *mountain snowy*, *hill*, *boardwalk*, *swamp*, *water cascade*, *ocean*, *coast* and *sky*. The number of images in each category ranges from 50 to 100. Figure 6 shows some example images. We can see that some categories are similar (*e.g. ocean* and *coast*, *boardwalk* and *swamp*, etc), while others are largely different. To derive the image representation, PCA-SIFT [8] descriptors are densely extracted from each training image, and then pooled together and quantized using K-means into 512 visual words. In this way, each image can be represented as a histogram of 512 bins.

All methods mentioned above are compared. For M-LDP, we introduce a global latent DP to capture common topics, with $q$-values set uniformly to 0.5, and a set of local latent DPs, each for a category. The prior probability of inheriting from the corresponding latent DP is 1.0, and that from other local DPs is 0.2. Whereas no prior knowledge about the similarity between categories is assumed, the latent DPs incorporated in this way still provide a mechanism for local coupling. For SNΓP, we use 28 latent DPs, each connected to a pair of categories. Again, we divide the data into two disjoint halves, respectively for training and testing, and evaluate the performance in terms of perplexity. The results are shown in Figure 7, where we can observe trends similar to those that we have seen on the NIPS data: local coupling helps model estimation, and our model under the M-LDP setting further reduces the perplexity (from 37 to 31, as compared to SNΓP). This is due to the more flexible way to configure local coupling that allows the weights of latent DPs to vary.

## 6  Conclusion

We have presented a principled approach to modeling grouped data, where mixture models for different groups are coupled via a set of latent DPs. The proposed framework allows each mixture model to inherit from multiple latent DPs, and each latent DP to contribute differently to different groups, thus providing great flexibility for model design. The experiments on both document analysis and image modeling has clearly demonstrated the utility of such flexibility. Particularly, the proposed method makes it possible to make various modeling choices, *e.g.* the use of latent DPs with different connection patterns, substantially improving the effectiveness of the estimated models. While $q$-values are treated as design parameters, it should be possible to extend this framework to incorporate prior models over these and other parameters. Such extensions should lead to constructions with richer structure capable of addressing more complex problems.

**Acknowledgements**

This research was partially supported by the Office of Naval Research Multidisciplinary Research Initiative (MURI) program, award N000141110688 and by DARPA award FA8650-11-1-7154.

# References

[1] David Blei and John Lafferty. Correlated topic models. In *Proc. of NIPS'06*, 2006.

[2] David M. Blei, Andrew Y. Ng, and Michael I. Jordan. Latent Dirichlet Allocation. *Journal of Machine Learning Research*, 3:993–1022, 2003.

[3] Yeonseung Chung and David B. Dunson. The local Dirichlet Process. *Annals of the Inst. of Stat. Math.*, 63(1):59–80, 2009.

[4] Li Fei-fei. A bayesian hierarchical model for learning natural scene categories. In *Proc. of CVPR'05*, 2005.

[5] Amir Globerson, Gal Chechik, Fernando Pereira, and Naftali Tishby. Euclidean embedding of co-occurrence data. *JMLR*, 8, 2007.

[6] J. E Griffin and M. F. J Steel. Order-Based Dependent Dirichlet Processes. *Journal of the American Statistical Association*, 101(473):179–194, March 2006.

[7] Thomas Hofmann. Probabilistic latent semantic indexing. In *Proc. of ACM SIGIR'99*, 1999.

[8] Yan Ke and Rahul Sukthankar. Pca-sift: A more distinctive representation for local image descriptors. In *Proc. of CVPR'04*, 2004.

[9] Seyoung Kim and Padhraic Smyth. Hierarchical dirichlet processes with random effects. In *Proc. of NIPS'06*, 2006.

[10] Jyri J. Kivinen, Erik B. Sudderth, and Michael I. Jordan. Learning multiscale representations of natural scenes using dirichlet processes. In *Proc. of CVPR'07*, 2007.

[11] S. Lazebnik, C. Schmid, and J. Ponce. Beyond bags of features: Spatial pyramid matching for recognizing natural scene categories. In *Proc. of CVPR'06*, 2006.

[12] Dahua Lin, Eric Grimson, and John Fisher. Construction of dependent dirichlet processes based on poisson processes. In *Advances of NIPS'10*, 2010.

[13] Steven N. MacEachern. Dependent Nonparametric Processes. In *Proceedings of the Section on Bayesian Statistical Science*, 1999.

[14] Radford M. Neal. Markov Chain Sampling Methods for Dirichlet Process Mixture Models. *Journal of computational and graphical statistics*, 9(2):249–265, 2000.

[15] Vinayak Rao and Yee Whye Teh. Spatial Normalized Gamma Processes. In *Proc. of NIPS'09*, 2009.

[16] Carl Edward Rasmussen. The Infinite Gaussian Mixture Model. In *Proc. of NIPS'00*, 2000.

[17] Lu Ren, David B. Dunson, and Lawrence Carin. The Dynamic Hierarchical Dirichlet Process. In *Proc. of ICML'08*, New York, New York, USA, 2008. ACM Press.

[18] J. Sethuraman. A Constructive Definition of Dirichlet Priors. *Statistica Sinica*, 4(2):639–650, 1994.

[19] Erik B. Sudderth, Antonio Torralba, William Freeman, and Alan Willsky. Describing visual scenes using transformed dirichlet processes. In *Proc. of NIPS'05*, 2005.

[20] Yee Whye Teh, Michael I. Jordan, Matthew J. Beal, and David M. Blei. Hierarchical Dirichlet Processes. *Journal of the American Statistical Association*, 101(476):1566–1581, 2006.

[21] Chang Wang, David Blei, and Fei-Fei Li. Simultaneous image classification and annotation. In *Proc. of CVPR'09*, 2009.

[22] J. Xiao, J. Hays, K. Ehinger, A. Oliva, and A. Torralba. Sun database: Large-scale scene recognition from abbey to zoo. In *Proc. of CVPR'10*, 2010.

